# Regulator Discovery from Gene Expression Time Series of Malaria Parasites: a Hierarchical Approach

**José Miguel Hernández-Lobato**
Escuela Politécnica Superior
Universidad Autónoma de Madrid, Madrid, Spain
Josemiguel.hernandez@uam.es

**Tjeerd Dijkstra**
Leiden Malaria Research Group
LUMC, Leiden, The Netherlands
t.dijkstra@lumc.nl

**Tom Heskes**
Institute for Computing and Information Sciences
Radboud University Nijmegen, Nijmegen, The Netherlands
t.heskes@science.ru.nl

## Abstract

We introduce a hierarchical Bayesian model for the discovery of putative regulators from gene expression data only. The hierarchy incorporates the knowledge that there are just a few regulators that by themselves only regulate a handful of genes. This is implemented through a so-called spike-and-slab prior, a mixture of Gaussians with different widths, with mixing weights from a hierarchical Bernoulli model. For efficient inference we implemented expectation propagation. Running the model on a malaria parasite data set, we found four genes with significant homology to transcription factors in an amoebe, one RNA regulator and three genes of unknown function (out of the top ten genes considered).

## 1 Introduction

Bioinformatics provides a rich source for the application of techniques from machine learning. Especially the elucidation of regulatory networks underlying gene expression has lead to a cornucopia of approaches: see [1] for review. Here we focus on one aspect of network elucidation, the identification of the regulators of the causative agent of severe malaria, *Plasmodium falciparum*. Several properties of the parasite necessitate a tailored algorithm for regulator identification:

- In most species gene regulation takes place at the first stage of gene expression when a DNA template is transcribed into mRNA. This transcriptional control is mediated by specific transcription factors. Few specific transcription factors have been identified in Plasmodium based on sequence homology with other species [2, 3]. This could be due to Plasmodium possessing a unique set of transcription factors or due to other mechanisms of gene regulation, e.g. at the level of mRNA stability or post-transcritional regulation.

- Compared with yeast, gene expression in Plasmodium is hardly changed by perturbations e.g. by adding chemicals or changing temperature [4]. The biological interpretation of this finding is that the parasite is so narrowly adapted to its environment inside a red blood cell that it follows a stereotyped gene expression program. From a machine learning point of view, this finding means that network elucidation techniques relying on perturbations of gene expression cannot be used.

- Similar to yeast [5], data for three different strains of the parasite with time series of gene expression are publicly available [6]. These assay all of Plasmodium's 5,600 genes for about 50 time points. In contrast to yeast, there are no ChIP-chip data available and fewer then ten transcription factor binding motifs are known.

Together, these properties point to a vector autoregressive model making use of the gene expression time series. The model should not rely on sequence homology information but it should be flexible enough to integrate sequence information in the future. This points to a Bayesian model as favored approach.

## 2 The model

We start with a semi-realistic model of transcription based on Michaelis-Menten kinetics [1] and subsequently simplify to obtain a linear model. Denoting the concentration of a certain mRNA transcript at time $t$ by $z(t)$ we write:

$$\frac{dz(t)}{dt} = \frac{V_1 a_1(t)^{M_1}}{K_1 + a_1(t)^{M_1}} \cdots \frac{V_N a_N(t)^{M_N}}{K_N + a_N(t)^{M_N}} \, p(t) - \frac{1}{\tau_z} z(t), \qquad (1)$$

with $a_j(t)$ the concentration of the j-th activator (positive regulator), $p(t)$ the concentration of RNA polymerase and $V_j$, $K_j$, $M_j$ and $\tau_z$ reaction constants. $N$ denotes the number of potential activators. The activator is thought to bind to DNA motifs upstream of the transcription start site and binds RNA polymerase which reads the DNA template to produce an mRNA transcript. $M_j$ can be thought of as the multiplicity of the motif, $\tau_z$ captures the characteristic life time of the transcript. While reasonably realistic, this equation harbors too many unknowns for reliable inference: $3N + 1$ with $N \approx 1000$. We proceed with several simplifications:

- $a_j(t) \ll K_j$: activator concentration is low;
- $p(t) = p_0$ is constant;
- $\frac{dz(t)}{dt} \approx \frac{z(t+\Delta) - z(t)}{\Delta}$ with $\Delta$ the sampling period;
- $\Delta \approx \tau_z$: sampling period roughly equal to transcript life time.

Counting time in units of $\Delta$ and taking logarithms on both sides, Equation (1) then simplifies to

$$\log z(t+1) = C + M_1 \log a_1(t) + \cdots + M_N \log a_N(t),$$

with $C = \log(TV_1 \cdots V_N p_0 / (K_1 \cdots K_N))$. This is a linear model for gene expression level given the expression levels of a set of activators. With a similar derivation one can include repressors [1].

### 2.1 A Bayesian model for sparse linear regression

Let $\mathbf{y}$ be a vector with the log expression of the target gene and $\mathbf{X} = (\mathbf{x}_1, \ldots, \mathbf{x}_N)$ a matrix whose columns contain the log expression of the candidate regulators. Assuming that the measurements are corrupted with additive Gaussian noise, we get $\mathbf{y} \sim \mathcal{N}(\mathbf{X}\boldsymbol{\beta}, \sigma^2 \mathbf{I})$ where $\boldsymbol{\beta} = (\beta_1, \ldots, \beta_N)^{\mathrm{T}}$ is a vector of regression coefficients and $\sigma^2$ is the variance of the noise. Such a linear model is commonly used [7, 8, 9]. Both $\mathbf{y}$ and $\mathbf{x}_1, \ldots, \mathbf{x}_N$ are mean-centered vectors with $T$ measurements. We specify an inverse gamma (IG) prior for $\sigma^2$ so that $\mathcal{P}(\sigma^2) = \mathrm{IG}(\sigma^2, \nu/2, \nu\lambda/2)$, where $\lambda$ is a prior estimate of $\sigma^2$ and $\nu$ is the sample size associated with that estimate. We assume that *a priori* all components $\beta_i$ are independent and take a so-called "spike and slab prior" [10] for each of them. That is, we introduce binary latent variables $\gamma_i$, with $\gamma_i = 1$ if $\mathbf{x}_i$ takes part in the regression of $\mathbf{y}$ and $\gamma_i = 0$ otherwise. Given $\boldsymbol{\gamma}$, the prior on $\boldsymbol{\beta}$ then reads

$$\mathcal{P}(\boldsymbol{\beta}|\boldsymbol{\gamma}) = \prod_{i=1}^{N} \mathcal{P}(\beta_i|\gamma_i) = \prod_{i=1}^{N} \mathcal{N}(\beta_i, 0, v_1)^{\gamma_i} \, \mathcal{N}(\beta_i, 0, v_0)^{1-\gamma_i} \,,$$

where $\mathcal{N}(x, \mu, \sigma^2)$ denotes a Gaussian density with mean $\mu$ and variance $\sigma^2$ evaluated at $x$. In order to enforce sparsity, the variance $v_1$ of the slab should be larger than the variance $v_0$ of the spike. Instead of picking the hyperparameters $v_1$ and $v_0$ directly, it is convenient to pick a threshold of practical significance $\delta$ so that $\mathcal{P}(\gamma_i = 1)$ gets more weight when $|\beta_i| > \delta$ and $\mathcal{P}(\gamma_i = 0)$ gets more weight when $|\beta_i| < \delta$ [10]. In this way, given $\delta$ and one of $v_1$ or $v_0$, we pick the other one such that

$$\delta^2 = \frac{\log(v_1/v_0)}{v_0^{-1} - v_1^{-1}} \,. \qquad (2)$$

Finally, we assign independent *Bernoulli* priors to the components of the latent vector $\boldsymbol{\gamma}$:

$$\mathcal{P}(\boldsymbol{\gamma}) = \prod_{i=1}^{N} \text{Bern}(\gamma_i, w) = \prod_{i=1}^{N} w^{\gamma_i} (1 - w)^{1 - \gamma_i} \,,$$

so that each of the $\mathbf{x}_1, \ldots, \mathbf{x}_N$ can independently take part in the regression with probability $w$. We can identify the candidate genes whose expression is more likely to be correlated with the target gene by means of the posterior distribution of $\boldsymbol{\gamma}$:

$$\mathcal{P}(\boldsymbol{\gamma}|\mathbf{y}, \mathbf{X}) = \int_{\boldsymbol{\beta}, \sigma^2} \mathcal{P}(\boldsymbol{\gamma}, \boldsymbol{\beta}, \sigma^2 | \mathbf{y}, \mathbf{X}) \, d\boldsymbol{\beta} \, d\sigma^2 \propto \int_{\boldsymbol{\beta}, \sigma^2} \mathcal{P}(\boldsymbol{\gamma}, \boldsymbol{\beta}, \sigma^2, \mathbf{y}|\mathbf{X}) \, d\boldsymbol{\beta} \, d\sigma^2 \,,$$

where

$$
\begin{aligned}
\mathcal{P}(\boldsymbol{\gamma}, \boldsymbol{\beta}, \sigma^2, \mathbf{y}|\mathbf{X}) &= \mathcal{N}(\mathbf{y}, \mathbf{X}\boldsymbol{\beta}, \sigma^2\mathbf{I}) \mathcal{P}(\boldsymbol{\beta}|\boldsymbol{\gamma}) \mathcal{P}(\boldsymbol{\gamma}) \mathcal{P}(\sigma^2) \\
&= \left[ \prod_{t=1}^{T} \mathcal{N}(y_t, \sum_{i=1}^{N} x_{i,t}\beta_i, \sigma^2) \right] \left[ \prod_{i=1}^{N} \left\{ \mathcal{N}(\beta_i, 0, v_1)^{\gamma_i} \, \mathcal{N}(\beta_i, 0, v_0)^{1-\gamma_i} \right\} \right] \\
&\quad \left[ \prod_{i=1}^{N} \text{Bern}(\gamma_i, w) \right] \text{IG}(\sigma^2, \nu/2, \nu\lambda/2) \,.
\end{aligned}
$$
(3)

Unfortunately, this posterior distribution cannot be computed exactly if the number $N$ of candidate genes is larger than 25. An approximation based on Markov Chain Monte Carlo (MCMC) methods has been proposed in [11].

## 2.2 A hierarchical model for gene regulation

In the section above we made use of the prior information that a target gene is typically regulated by a small number of regulators. We have not yet made use of the prior information that a regulator typically regulates more than one gene. We incorporate this information by a hierarchical extension of our previous model. We introduce a vector $\boldsymbol{\tau}$ of binary latent variables where $\tau_i = 1$ if gene $i$ is a regulator and $\tau_i = 0$ otherwise. The following joint distribution captures this idea:

$$
\begin{aligned}
\mathcal{P}(\boldsymbol{\tau}, \boldsymbol{\gamma}, \boldsymbol{\beta}, \boldsymbol{\sigma^2}|\mathbf{X}) &= \left[ \prod_{j=1}^{N} \prod_{t=1}^{T-1} \mathcal{N}(x_{j,t+1}, \sum_{i=1, i\neq j}^{N} x_{i,t}\beta_{j,i}, \sigma_j^2) \right] \\
&\quad \left[ \prod_{j=1}^{N} \prod_{i=1, i\neq j}^{N} \mathcal{N}(\beta_{j,i}, 0, v_1)^{\gamma_{j,i}} \, \mathcal{N}(\beta_{j,i}, 0, v_0)^{1-\gamma_{j,i}} \right] \left[ \prod_{j=1}^{N} \text{IG}(\sigma_j^2, \nu_j/2, \nu_j\lambda_j/2) \right] \\
&\quad \left[ \prod_{j=1}^{N} \prod_{i=1, i\neq j}^{N} \text{Bern}(\gamma_{j,i}, w_1)^{\tau_i} \, \text{Bern}(\gamma_{j,i}, w_0)^{1-\tau_i} \right] \left[ \prod_{i=1}^{N} \text{Bern}(\tau_i, w) \right] \,.
\end{aligned}
$$
(4)

In this hierarchical model, $\boldsymbol{\gamma}$ is a matrix of binary latent variables where $\gamma_{j,i} = 1$ if gene $i$ takes part in the regression of gene $j$ and $\gamma_{j,i} = 0$ otherwise. The relationship between regulators and regulatees suggests that $\mathcal{P}(\gamma_{j,i} = 1|\tau_i = 1)$ should be bigger than $\mathcal{P}(\gamma_{j,i} = 1|\tau_i = 0)$ and thus $w_1 > w_0$. Matrix $\boldsymbol{\beta}$ contains regression coefficients where $\beta_{j,i}$ is the regression coefficient between the expression of gene $i$ and the delayed expression of gene $j$. Hyperparameter $w$ represents the prior probability of any gene being a regulator and the elements $\sigma_j^2$ of the vector $\boldsymbol{\sigma^2}$ contain the variance of the noise in each of the $N$ regressions. Hyperparameters $\lambda_j$ and $\nu_j$ have the same meaning as in the model for sparse linear regression. The corresponding plate model is illustrated in Figure 1.

We can identify the genes more likely to be regulators by means of the posterior distribution $\mathcal{P}(\boldsymbol{\tau}|\mathbf{X})$. Compared with the sparse linear regression model we expanded the number of latent variables from $\mathcal{O}(N)$ to $\mathcal{O}(N^2)$. In order to keep inference feasible we turn to an approximate inference technique.

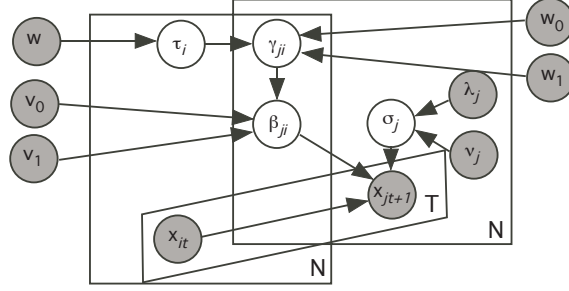

Figure 1: The hierarchical model for gene regulation.

## 3 Expectation propagation

The Expectation Propagation (EP) algorithm [12] allows to perform approximate Bayesian inference. In all Bayesian problems, the joint distribution of the model parameters $\boldsymbol{\theta}$ and a data set $\mathcal{D} = \{(\mathbf{x}_i, y_i) : i = 1, \ldots, n\}$ with i.i.d. elements can be expressed as a product of terms

$$\mathcal{P}(\boldsymbol{\theta}, \mathcal{D}) = \prod_{i=1}^{n} \mathcal{P}(y_i|\mathbf{x}_i, \boldsymbol{\theta})\mathcal{P}(\boldsymbol{\theta}) = \prod_{i=1}^{n+1} t_i(\boldsymbol{\theta}), \tag{5}$$

where $t_{n+1}(\boldsymbol{\theta}) = \mathcal{P}(\boldsymbol{\theta})$ is the prior distribution for $\boldsymbol{\theta}$ and $t_i(\boldsymbol{\theta}) = \mathcal{P}(y_i|\mathbf{x}_i, \boldsymbol{\theta})$ for $i = 1, \ldots, n$. Expectation propagation proceeds to approximate (5) with a product of simpler terms

$$\prod_{i=1}^{n+1} t_i(\boldsymbol{\theta}) \approx \prod_{i=1}^{n+1} \tilde{t}_i(\boldsymbol{\theta}) = \mathcal{Q}(\boldsymbol{\theta}), \tag{6}$$

where all the term approximations $\tilde{t}_i$ are restricted to belong to the same family $\mathcal{F}$ of exponential distributions, but they do not have to integrate 1. Note that $\mathcal{Q}$ will also be in $\mathcal{F}$ because $\mathcal{F}$ is closed under multiplication. Each term approximation $\tilde{t}_i$ is chosen so that

$$\mathcal{Q}(\boldsymbol{\theta}) = \tilde{t}_i(\boldsymbol{\theta}) \prod_{j \neq i} \tilde{t}_j(\boldsymbol{\theta}) = \tilde{t}_i(\boldsymbol{\theta})\mathcal{Q}^{\backslash i}(\boldsymbol{\theta})$$

is as close as possible to

$$t_i(\boldsymbol{\theta}) \prod_{j \neq i} \tilde{t}_j(\boldsymbol{\theta}) = t_i(\boldsymbol{\theta})\mathcal{Q}^{\backslash i}(\boldsymbol{\theta}),$$

in terms of the direct Kullback-Leibler (K-L) divergence. The pseudocode of the EP algorithm is:

1. Initialize the term approximations $\tilde{t}_i$ and $\mathcal{Q}$ to be uniform.
2. Repeat until all $\tilde{t}_i$ converge:
    (a) Choose a $\tilde{t}_i$ to refine and remove it from $\mathcal{Q}$ to get $\mathcal{Q}^{\backslash i}$ (e.g. dividing $\mathcal{Q}$ by $\tilde{t}_i$).
    (b) Update the term $\tilde{t}_i$ so that it minimizes the K-L divergence between $t_i\mathcal{Q}^{\backslash i}$ and $\tilde{t}_i\mathcal{Q}^{\backslash i}$.
    (c) Re-compute $\mathcal{Q}$ so that $\mathcal{Q} = \tilde{t}_i\mathcal{Q}^{\backslash i}$.

The optimization problem in step (b) is solved by matching sufficient statistics between a distribution $\mathcal{Q}'$ within the $\mathcal{F}$ family and $t_i\mathcal{Q}^{\backslash i}$, the new $\tilde{t}_i$ is then equal to $\mathcal{Q}'/\mathcal{Q}^{\backslash i}$. Because $\mathcal{Q}$ belongs to the exponential family it is generally trivial to calculate its normalization constant. Once $\mathcal{Q}$ is normalized it can approximate $\mathcal{P}(\boldsymbol{\theta}|\mathcal{D})$. Finally, EP is not guaranteed to converge, although convergence can be improved by means of damped updates or double-loop algorithms [13].

### 3.1 EP for sparse linear regression

The application of EP to the models of Section 2 introduces some nontrivial technicalities. Furthermore, we describe several techniques to speed up the EP algorithm. We approximate $\mathcal{P}(\boldsymbol{\gamma}, \boldsymbol{\beta}, \sigma^2, \mathbf{y}|\mathbf{X})$ for sparse linear regression by means of a factorized exponential distribution:

$$\mathcal{P}(\boldsymbol{\gamma}, \boldsymbol{\beta}, \sigma^2, \mathbf{y}|\mathbf{X}) \approx \left[\prod_{i=1}^{N} \text{Bern}(\gamma_i, q_i)\mathcal{N}(\beta_i, \mu_i, s_i)\right] \text{IG}(\sigma^2, a, b) \equiv \mathcal{Q}(\boldsymbol{\gamma}, \boldsymbol{\beta}, \sigma^2), \tag{7}$$

where $\{q_i, \mu_i, s_i \,:\, i = 1, \ldots, N\}$, $a$ and $b$ are free parameters. Note that in the approximation $\mathcal{Q}(\boldsymbol{\gamma}, \boldsymbol{\beta}, \sigma^2)$ all the components of the vectors $\boldsymbol{\gamma}$ and $\boldsymbol{\beta}$ and the variable $\sigma^2$ are considered to be independent; this allows the approximation of $\mathcal{P}(\boldsymbol{\gamma}|\mathbf{y}, \mathbf{X})$ by $\prod_{i=1}^n \mathrm{Bern}(\gamma_i, q_i)$. We tune the parameters of $\mathcal{Q}(\boldsymbol{\gamma}, \boldsymbol{\beta}, \sigma^2)$ by means of EP over the unnormalized density $\mathcal{P}(\boldsymbol{\gamma}, \boldsymbol{\beta}, \sigma^2, \mathbf{y}|\mathbf{X})$. Such density appears in (3) as a product of $T + N$ terms (not counting the priors) which correspond to the $t_i$ terms in (5). This way, we have $T + N$ term approximations with the same form as (7) and which correspond to the term approximations $\tilde{t}_i$ in (6). The complexity is $\mathcal{O}(TN)$ per iteration, because updating any of the first $T$ term approximations requires $N$ operations. However, some of the EP update operations require to compute integrals which do not have a closed form expression. To avoid that, we employ the following simplifications when we update the first $T$ term approximations:

1. When updating the parameters $\{\mu_i, s_i \,:\, i = 1, \ldots, N\}$ of the Gaussians in the term approximations, we approximate a Student's $t$-distribution by means of a Gaussian distribution with the same mean and variance. This approximation becomes more accurate as the degrees of freedom of the $t$-distribution increase.

2. When updating the parameters $\{a, b\}$ of the IG in the term approximations, instead of propagating the sufficient statistics of an IG distribution we propagate the expectations of $1/\sigma^2$ and $1/\sigma^4$. To achieve this, we have to perform two approximations like the one stated above. Note that in this case we are not minimizing the direct K-L divergence. However, at convergence, we expect the resulting IG in (7) to be sufficiently accurate.

In order to improve convergence, we re-update all the $N$ last term approximations each time one of the first $T$ term approximations is updated. Computational complexity does not get worse than $\mathcal{O}(TN)$ and the resulting algorithm turns out to be faster. By comparison, the MCMC method in [11] takes $\mathcal{O}(N^2)$ steps to generate a single sample from $\mathcal{P}(\boldsymbol{\gamma}|\mathbf{y}, \mathbf{X})$. On problems of much smaller size than we will consider in our experiments, one typically requires on the order of 10000 samples to obtain reasonably accurate estimates [10].

## 3.2 EP for gene regulation

We approximate $\mathcal{P}(\boldsymbol{\tau}, \boldsymbol{\gamma}, \boldsymbol{\beta}, \boldsymbol{\sigma^2}|\mathbf{X})$ by the factorized exponential distribution

$$
\mathcal{Q}(\boldsymbol{\tau}, \boldsymbol{\gamma}, \boldsymbol{\beta}, \boldsymbol{\sigma^2}) \;=\; \left[ \prod_{j=1}^{N} \prod_{i=1,i \neq j}^{N} \mathrm{Bern}(\gamma_{j,i}, w_{j,i}) \right] \left[ \prod_{i=1}^{N} \mathrm{Bern}(\tau_i, t_i) \right]
$$
$$
\left[ \prod_{j=1}^{N} \prod_{i=1,i \neq j}^{N} \mathcal{N}(\beta_{j,i}, \mu_{j,i}, s_{j,i}) \right] \left[ \prod_{j=1}^{N} \mathrm{IG}(\sigma_j^2, a_j, b_j) \right] ,
$$

where $\{a_j, b_j, t_i, w_{j,i}, \mu_{j,i}, s_{j,i} \,:\, i = 1, \ldots, N \,;\, j = 1, \ldots, N \,;\, i \neq j\}$ are free parameters. The posterior probability $\mathcal{P}(\boldsymbol{\tau}|\mathbf{X})$ that indicates which genes are more likely to be regulators can then be approximated by $\prod_{i=1}^{N} \mathrm{Bern}(\tau_i, t_i)$. Again, we fix the parameters in $\mathcal{Q}(\boldsymbol{\tau}, \boldsymbol{\gamma}, \boldsymbol{\beta}, \boldsymbol{\sigma^2})$ by means of EP over the joint density $\mathcal{P}(\boldsymbol{\tau}, \boldsymbol{\gamma}, \boldsymbol{\beta}, \boldsymbol{\sigma^2}|\mathbf{X})$. It is trivial to adapt the EP algorithm used in the sparse linear regression model to this new case: the terms to be approximated are the same as before except for the new $N(N-1)$ terms for the prior on $\boldsymbol{\gamma}$. As in the previous section and in order to improve convergence, we re-update all the $N(N-1)$ term approximations corresponding to the prior on $\boldsymbol{\beta}$ each time $N$ of the $N(T-1)$ term approximations corresponding to regressions are updated. In order to reduce memory requirements, we associate all the $N(N-1)$ terms for the prior on $\boldsymbol{\beta}$ into a single term, which we can do because they are independent so that we only store in memory one term approximation instead of $N(N-1)$. We also group the $N(N-1)$ terms for the prior on $\boldsymbol{\gamma}$ into $N$ independent terms and the $N(T-1)$ terms for the regressions into $T-1$ independent terms. Assuming a constant number of iterations (in our experiments, we need at most 20 iterations for EP to converge), the computational complexity and the memory requirements of the resulting algorithm are $\mathcal{O}(TN^2)$. This indicates that it is feasible to analyze data sets which contain the expression pattern of thousands of genes. An MCMC algorithm would require $\mathcal{O}(N^3)$ to generate just a single sample.

# 4 Experiments with artificial data

We carried out experiments with artificially generated data in order to validate the EP algorithms. In the experiments for sparse linear regression we fixed the hyperparameters in (3) so that $\nu = 3$, $\lambda$ is the sample variance of the target vector $\mathbf{y}$, $v_1 = 1$, $\delta = N^{-1}$, $v_0$ is chosen according to (2) and $w = N^{-1}$. In the experiment for gene regulation we fixed the hyperparameters in (4) so that $w = (N-1)^{-1}$, $\nu_i = 3$ and $\lambda_i$ is the sample variance of the vector $\mathbf{x}_i$, $w_1 = 10^{-1}(N-1)^{-1}$, $w_0 = 10^{-2}(N-1)^{-1}$, $v_1 = 1$, $\delta = 0.2$ and $v_0$ is chosen according to (2). Although the posterior probabilities are sensitive to some of the choices, the orderings of these probabilities, e.g., to determine the most likely regulators, are robust to even large changes.

## 4.1 Sparse linear regression

In the first experiment we set $T = 50$ and generated $\mathbf{x}_1, \ldots, \mathbf{x}_{6000} \sim \mathcal{N}(0, 3^2\mathbf{I})$ candidate vectors and a target vector $\mathbf{y} = \mathbf{x}_1 - \mathbf{x}_2 + 0.5\,\mathbf{x}_3 - 0.5\,\mathbf{x}_4 + \varepsilon$, where $\varepsilon \sim \mathcal{N}(0, \mathbf{I})$. The EP algorithm assigned values close to 1 to $w_1$ and $w_2$, the parameters $w_3$ and $w_4$ obtained values $5.2 \cdot 10^{-3}$ and $0.5$ respectively and $w_5, \ldots, w_{6000}$ were smaller than $3 \cdot 10^{-4}$. We repeated the experiment several times (each time using new data) and obtained similar results on each run.

In the second experiment we set $T = 50$ and generated a target vector $\mathbf{y} \sim \mathcal{N}(0, 3^2\mathbf{I})$ and $\mathbf{x}_1, \ldots, \mathbf{x}_{500}$ candidate vectors so that $\mathbf{x}_i = \mathbf{y} + \varepsilon_i$ for $i = 2, \ldots, 500$, where $\varepsilon_i \sim \mathcal{N}(0, \mathbf{I})$. The candidate vector $\mathbf{x}_1$ is generated as $\mathbf{x}_1 = \mathbf{y} + 0.5\,\varepsilon_1$ where $\varepsilon_1 \sim \mathcal{N}(0, \mathbf{I})$. This way, the noise in $\mathbf{x}_1$ is twice as small as the noise in the other candidate vectors. Note that all the candidate vectors are highly correlated with each other and with the target vector. This is what happens in gene expression data sets where many genes show similar expression patterns. We ran the EP algorithm 100 times (each time using new data) and it always assigned to all the $w_1, \ldots, w_{500}$ more or less the same value of $6 \cdot 10^{-4}$. However, $w_1$ obtained the highest value on 54 of the runs and it was among the three $w$s with highest value on 87 of the runs.

Finally, we repeated these experiments setting $N = 100$, using the MCMC method of [11] and the EP algorithm for sparse linear regression. Both techniques produced results that are statistically indistinguishable (the approximations obtained through EP fall within the variation of the MCMC method), for EP within a fraction of the time of MCMC.

## 4.2 Gene regulation

In this experiment we set $T = 50$ and generated a vector $\mathbf{z}$ with $T + 1$ values from a sinusoid. We then generated 49 more vectors $\mathbf{x}_2, ..., \mathbf{x}_{50}$ where $x_{i,t} = z_t + \varepsilon_{i,t}$ for $i = 2, \ldots, 50$ and $t = 1, \ldots, T$, where $\varepsilon_{i,t} \sim \mathcal{N}(0, \sigma^2)$ and $\sigma$ is one fourth of the sample standard deviation of $\mathbf{z}$. We also generated a vector $\mathbf{x}_1$ so that $x_{1,t} = z_{t+1} + \varepsilon_t$ where $t = 1, \ldots, T$ and $\varepsilon_t \sim \mathcal{N}(0, \sigma^2)$. In this way, $\mathbf{x}_1$ acts as a regulator for $\mathbf{x}_2, ..., \mathbf{x}_{50}$. A single realization of the vectors $\mathbf{x}_1, \ldots, \mathbf{x}_{50}$ is displayed on the left of Figure 2. We ran the EP algorithm for gene regulation over 100 different realizations of $\mathbf{x}_1, \ldots, \mathbf{x}_{50}$. The algorithm assigned $t_1$ the highest value on 33 of the runs and $\mathbf{x}_1$ was ranked among the top five on 74 of the runs. This indicates that the EP algorithm can successfully detect small differences in correlations and should be able to find new regulators in real microarray data.

# 5 Experiments with real microarray data

We applied our algorithm to four data sets. The first is a yeast cell-cycle data set from [5] which is commonly used as a benchmark for regulator discovery. Data sets two through four are from three different Plasmodium strains [6]. Missing values were imputed by nearest neighbors [14] and the hyperparameters were fixed at the same values as in Section 4. The yeast cdc15 data set contains 23 measurements of 6178 genes. We singled out 751 genes which met a minimum criterion for cell cycle regulation [5]. The top ten genes with the highest values for $\tau$ along with their annotation from the *Saccharomyces* Genome database are listed in table 5: the top two genes are specific transcription factors and IOC2 is associated with transcription regulation. As 4% of the yeast genome is associated with transcription the probability of this occurring by chance is 0.0062. However, although the result is statistically significant, we were disappointed to find none of the known cell-cycle regulators (like ACE2, FKH* or SWI*) among the top ten.

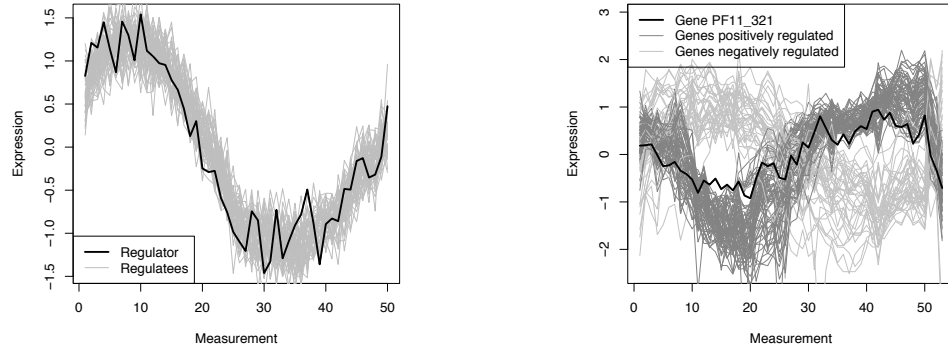

Figure 2: Left: Plot of the vectors $\mathbf{x}_2, ..., \mathbf{x}_{50}$ in grey and the vector $\mathbf{x}_1$ in black. The vector $\mathbf{x}_1$ contains the expression of a regulator which would determine the expressions in $\mathbf{x}_2, ..., \mathbf{x}_{50}$. Right: Expressions of gene PF11_321 (black) and the 100 genes which are more likely to be regulated by it (light and dark grey). Two clusters of positively and negatively regulated genes can be appreciated.

| rank | standard name | common name | annotation |
|---|---|---|---|
| 1 | YLR098c | CHA4 | DNA binding transcriptional activator |
| 2 | YOR315w | SFG1 | putative transcription factor for growth of superficial pseudohyphae |
| 3 | YJL073w | JEM1 | DNAJ-like chaperone |
| 4 | YOR023c | AHC1 | subunit of the ADA histone acetyl transferase complex |
| 5 | YOR105w | - | dubious open reading frame |
| 6 | YLR095w | IOC2 | transcription elongation |
| 7 | YOR321w | PMT3 | protein O-mannosyl transferase |
| 8 | YLR231c | BNA5 | kynureninase |
| 9 | YOR248w | - | dubious open reading frame |
| 10 | YOR247w | SRL1 | mannoprotein |

The three data sets for the malaria parasite [6] contain 53 measurements (3D7), 50 measurements (Dd2) and 48 measurements (HB3). We focus on 3D7 as this is the sequenced reference strain. We singled out 751 genes who showed the highest variation as quantified by the interquartile range of the expression measurements. The top ten genes with the highest values for $\tau$ along with their annotation from PlasmoDB are listed in table 5. Recalling the motivation for our approach, the paucity of known transcription factors, we cannot expect to find many annotated regulators in PlasmoDB version 5.4. Thus, we list the BLASTP hits provided by PlasmoDB instead of the absent annotation. These hits were the highest scoring ones outside of the genus Plasmodium. We find four genes with a large identity to transcription factors in Dictyostelium (a recently sequenced social amoebe) and one annotated helicase which typically functions in post-transcriptional regulation. Interestingly three genes have no known function and could be regulators.

| rank | standard name | annotation or selected BLASTP hits |
|---|---|---|
| 1 | PFC0950c | 25% identity to GATA binding TF in Dictyostelium |
| 2 | PF11_0321 | 25% identity to putative WRKY TF in Dictyostelium |
| 3 | PFI1210w | no BLASTP matches outside Plasmodium genus |
| 4 | MAL6P1.233 | no BLASTP matches outside Plasmodium genus |
| 5 | PFD0175c | 32% identity to GATA binding TF in Dictyostelium |
| 6 | MAL7P1.34 | 35% identity to GATA binding TF in Dictyostelium |
| 7 | MAL6P1.182 | N-acetylglucosaminyl-phosphatidylinositol de-n-acetylase |
| 8 | PF13_0140 | dihydrofolate synthase/folylpolyglutamate synthase |
| 9 | PF13_0138 | no BLASTP matches outside Plasmodium genus |
| 10 | MAL13P1.14 | DEAD box helicase |

Results for the HB3 strain were similar in that five putative regulators were found. Somewhat disappointing, we found only one putative regulator (a helicase) among the top ten genes for Dd2.

# 6 Conclusion and discussion

Our approach enters a field full of methods enforcing sparsity ([15, 8, 7, 16, 9]). Our main contributions are: a hierarchical model to discover regulators, a tractable algorithm for fast approximate inference in models with many interacting variables, and the application to malaria.

Arguably most related is the hierarchical model in [15]. The covariates in this model are a dozen external variables, coding experimental conditions, instead of the hundreds of expression levels of other genes as in our model. Furthermore, the prior in [15] enforces sparsity on the "columns" of $\beta$ to implement the idea that some genes are not influenced by any of the experimental conditions. Our prior, on the other hand, enforces sparsity on the "rows" in order to find regulators.

Future work could include more involved priors, e.g., enforcing sparsity on both "rows" and "columns" or incorporating information from DNA sequence data. The approximate inference techniques described in this paper make it feasible to evaluate such extensions in a fraction of the time required by MCMC methods.

# References

[1] T.S. Gardner and J.J. Faith. Reverse-engineering transcription control networks. *Physics of Life Reviews*, 2:65–88, 2005.

[2] R. Coulson, N. Hall, and C. Ouzounis. Comparative genomics of transcriptional control in the human malaria parasite *Plasmodium falciparum*. *Genome Res.*, 14:1548–1554, 2004.

[3] S. Balaji, M.M. Babu, L.M. Iyer, and L. Aravind. Discovery of the principal specific transcription factors of apicomplexa and their implication for the evolution of the ap2-integrase dna binding domains. *Nucleic Acids Research*, 33(13):3994–4006, 2005.

[4] T. Sakata and E.A. Winzeler. Genomics, systems biology and drug development for infectuous diseases. *Molecular BioSystems*, 3:841–848, 2007.

[5] P.T. Spellman, G. Sherlock, V.R. Iyer, K. Anders, M.B. Eisen, P.O. Brown, and D. Botstein. Comprehensive identification of cell cycle-regulated genes of the yeast *Saccharomyces cerevisiae* by microarray hybridization. *Molecular Biology of the Cell*, 9(12):3273–3297, 1998.

[6] M. LLinas, Z. Bozdech, E. D. Wong, A.T. Adai, and J. L. DeRisi. Comparative whole genome transcriptome analysis of three *Plasmodium falciparum* strains. *Nucleic Acids Research*, 34(4):1166–1173, 2006.

[7] M. Beal. *Variational Algorithms for Approximate Bayesian Inference*. PhD thesis, UCL, 2003.

[8] C. Sabatti and G.M. James. Bayesian sparse hidden components analysis for transcription regulation networks. *Bioinformatics*, 22(6):739–746, 2006.

[9] S.T. Jensen, G. Chen, and C.J. Stoeckert. Bayesian variable selection and data integration for biological regulatory networks. *The Annals of Applied Statistics*, 1:612–633, 2007.

[10] E.I. George and R.E. McCulloch. Approaches for Bayesian variable selection. *Statistica Sinica*, 7:339–374, 1997.

[11] E.I. George and R.E. McCulloch. Variable selection via Gibbs sampling. *Journal of the American Statistical Association*, 88(423):881–889, 1993.

[12] T. Minka. *A family of algorithms for approximate Bayesian inference*. PhD thesis, MIT, 2001.

[13] T. Heskes and O. Zoeter. Expectation propagation for approximate inference in dynamic Bayesian networks. *In UAI-2002*, pages 216–223, 2002.

[14] O. Troyanskaya, M. Cantor, P. Brown, T. Hastie, R. Tibshirani, and D. Botstein. Missing value estimation methods for dna microarrays. *Bioinformatics*, 17(6):520–525, 2001.

[15] J. Lucas, C. Carvalho, Q. Wang, A. Bild, J. Nevins, and M. West. Sparse statistical modelling in gene expression genomics. In K.A. Do, P. Müller, and M. Vannucci, editors, *Bayesian inference for gene expression and proteomics*. Springer, 2006.

[16] M.Y. Park, T. Hastie, and R. Tibshirani. Averaged gene expressions for regression. *Biostatistics*, 8:212–227, 2007.

